# A recurrent model of the interaction between Prefrontal and Inferotemporal cortex in delay tasks

ALFONSO RENART, NÉSTOR PARGA
*Departamento de Física Teórica*
*Universidad Autónoma de Madrid*
*Canto Blanco, 28049 Madrid, Spain*
http://www.ft.uam.es/neurociencia/GRUPO/grupo_english.html
and
EDMUND T. ROLLS
*Oxford University*
*Department of Experimental Psychology*
*South Parks Road, Oxford OX1 3UD, England*

## Abstract

A very simple model of two reciprocally connected attractor neural networks is studied analytically in situations similar to those encountered in delay match-to-sample tasks with intervening stimuli and in tasks of memory guided attention. The model qualitatively reproduces many of the experimental data on these types of tasks and provides a framework for the understanding of the experimental observations in the context of the attractor neural network scenario.

## 1 Introduction

Working memory is usually defined as the capability to actively hold information in memory for short periods of time. In primates, visual working memory is usually studied in experiments in which, after the presentation of a given visual stimulus, the monkey has to withhold its response during a certain delay period in which no specific visual stimulus is shown. After the delay, another stimulus is presented and the monkey has to make a response which depends on the interaction between the two stimuli. In order to bridge the temporal gap between the stimuli, the first one has to be held in memory during the delay. Electrophysiological recordings in primates during the performance of this type of tasks has revealed that some populations of neurons in different brain areas such as prefrontal (PF), inferotemporal (IT) or posterior parietal (PP) cortex, maintain approximately constant firing rates during the delay periods (for a review see [1]) and this delay activity states have been postulated as the internal representations of the stimuli provoking them [2]. Although up to now most of the modeling effort regarding the operation of networks able to support stable delay activity states has been put in the study of uni-modular (homogeneous) networks, there is evidence that in order for the monkey to solve the tasks satisfactorily, the interaction of several different neural structures is needed. A number of studies of delay match-to-sample tasks with intervening stimuli in primates performed by Desimone and

colleagues has revealed that although IT cortex supports delay activity states and shows memory related effects (differential responses to the same, fixed stimulus depending on its status on the trial, e.g. whether it matches or not the sample), it cannot, by itself, provide the information necessary to solve the task, as the delay activity states elicited by each of the stimuli in a sequence are disrupted by the input information associated with each new stimulus presented [3, 4, 5]. Another structure is therefore needed to store the information for the whole duration of the trial. PF cortex is a candidate, since it shows selective delay activity maintained through entire trials even with intervening stimuli [6]. A series of parallel experiments by the same group on memory guided attention [7, 8] have also shown differential firing of IT neurons in response to the *same* visual stimulus shown after a delay (an array of figures), depending on previous information shown before the delay (one of the figures in the array working as a target stimulus). This evidence suggests a distributed memory system as the proper scenario to study working memory tasks as those described above. Taking into account that both IT and PF cortex are known to be able to support delay activity states, and that they are bi-directionally connected, in this paper we propose a simple model consisting of two reciprocally connected attractor neural networks to be identified with IT and PF cortex. Despite its simplicity, the model is able to qualitatively reproduce the behavior of IT and PF cortex during delay match-to-sample tasks with intervening stimuli, the behavior of IT cells during memory guided attention tasks, and to provide an unified picture of these experimental data in the context of associative memory and attractor neural networks.

## 2   Model and dynamics

The model network consists of a large number of (excitatory) neurons arranged in two modules. Following [9, 10], each neuron is assumed to be a dynamical element which transforms an incoming afferent current into an output spike rate according to a given transduction function. A given afferent current $I_{ai}$ to neuron $i$ $(i = 1, \ldots, N)$ in module $a$ $(a = \mathbf{IT}, \mathbf{PF})$ decays with a characteristic time constant $\mathcal{T}$ but increases proportionally to the spike rates $\nu_{bj}$ of the rest of the neurons in the network (both from inside and outside its module) connected to it, the contribution of each presynaptic neuron, e.g. neuron $j$ from module $b$, being proportional to the synaptic efficacy $J_{ij}^{ab}$ between the two. This can be expressed through the following equation

$$\frac{dI_{ai}(t)}{dt} = -\frac{I_{ai}(t)}{\mathcal{T}} + \sum_{bj} J_{ij}^{(a,b)} \nu_{bj} + h_{ai}^{(ext)} \ . \tag{1}$$

An external current $h_{ai}^{(ext)}$ from outside the network, representing the stimuli, can also be imposed on every neuron. Selective stimuli are modeled as proportional to the stored patterns, i.e. $h_{ai}^{\mu(ext)} = h_a \eta_{ai}^\mu$, where $h_a$ is the intensity of the external current to module $a$.

The transduction function of the neurons transforming currents into rates has been chosen as a threshold hyperbolic tangent of gain $G$ and threshold $\theta$.

The synaptic efficacies between the neurons of each module and between the neurons in different modules are respectively [11, 12]

$$J_{ij}^{(a,a)} = \frac{J_0}{f(1-f)N_t} \sum_{\mu=1}^{P} (\eta_{ai}^\mu - f)(\eta_{aj}^\mu - f) \quad i \neq j \ ; \quad a = \mathbf{IT}, \mathbf{PF} \tag{2}$$

$$J_{ij}^{(a,b)} = \frac{g}{f(1-f)N_t} \sum_{\mu=1}^{P} (\eta_{ai}^\mu - f)(\eta_{bj}^\mu - f) \quad \forall \ i,j \ ; \quad a \neq b \ . \tag{3}$$

The intra-modular connections express the learning of $P$ binary patterns $\{\eta^{\mu}_{ai} = 0, 1, \mu = 1, \ldots, P\}$ by each module, each of them signaling which neurons are active in each of the sustained activity configurations. Each variable $\eta^{\mu}_{ai}$ is supposed to take the values 1 and 0 with probabilities $f$ and $(1 - f)$ respectively, independently across neurons and across patterns. The inter-modular connections reflect the temporal associations between the sustained activity states of each module. In this way, every stored pattern $\mu$ in the IT module has an associated pattern in the PF module which is labelled by the same index. The normalization constant $N_t = N(J_0 + g)$ has been chosen so that the sum of the magnitudes of the inter- and the intra-modular connections remains constant and equal to 1 while their relative values are varied. When this constraint is imposed the strength of the connections can be expressed in terms of a single independent parameter $g$ measuring the relative intensity of the inter- vs. the intra-modular connections ($J_0$ can be set equal to 1 everywhere). We will limit our study to the case where the number of stored patterns per module $P$ does not increase proportionally to the size of the modules $N$ since a large number of stored patterns does not seem necessary to describe the phenomenology of the delay match-to-sample experiments.

Since the number of neurons in a typical network one may be interested in is very large, e.g. $\sim 10^5 - 10^6$, the analytical treatment of the set of coupled differential equations (1) becomes intractable. On the other hand, when the number of neurons is large, a reliable description of the asymptotic solutions of these equations can be found using the techniques of statistical mechanics [13, 9]. In this framework, instead of characterizing the states of the system by the state of every neuron, this characterization is performed in terms of *macroscopic* quantities called *order parameters* which measure and quantify some global properties of the network as a whole. The relevant order parameters appearing in the description of our system are the overlaps of the state of each module with each of the stored patterns $m^{\mu}_a$, defined as:

$$m^{\mu}_a = \frac{1}{\chi N} \ll \sum_i (\eta^{\mu}_{ai} - f)\nu_{ai} \gg_\eta \; , \tag{4}$$

where the symbol $\ll \ldots \gg_\eta$ stands for an average over the stored patterns.

Using the free energy per neuron of the system at zero temperature $\mathcal{F}$ (which we do not write explicitly to reduce the technicalities to a minimum) we have modeled the experiments by giving the order parameters the following dynamics:

$$\mathcal{T} \frac{\partial m^{\mu}_a}{\partial t} = -\frac{\partial \mathcal{F}}{\partial m^{\mu}_a} \; . \tag{5}$$

This dynamics ensures that the stationary solutions, corresponding to the values of the order parameters at the attractors, correspond also to minima of the free energy, and that, as the system evolves, the free energy is always minimized through its gradient. The time constant of the macroscopic dynamics is a free parameter which has been chosen equal to the time constant of the individual neurons, reflecting the assumption that neurons operate in parallel. Its value has been set to $\mathcal{T} = 10 \; ms$. Equations (5) have been solved by a simple discretizing procedure (first order Runge-Kutta method).

Since not all neurons in the network receive the same inputs, not all of them behave in the same way, i.e. have the same firing rates. In fact, the neurons in each of the module can be split into different sub-populations according to their state of activity in each of the stored patterns. The mean firing rate of the neurons in each sub-population depends on the particular state realized by the network (characterized by the values of the order parameters). Associated to each pattern there are two larger sub-populations, to be denoted as foreground (all active neurons) and background (all inactive neurons) of that pattern.

The overlap with a given pattern can be expressed as the difference between the mean firing rate of the neurons in its foreground and its background. The average is performed over all other sub-populations to which each neuron in the foreground (background) may belong to, where the probability of a given sub-population is equal to the fraction of neurons in the module belonging to it (determined by the probability distribution of the stored patterns as given above). This partition of the neurons into sub-populations is appealing since, in experiments, cells are usually classified in terms of their response properties to a set of fixed stimuli, i.e. whether each stimulus is effective or ineffective in driving their response.

The modeling of the different experiments proceeded according to the macroscopic dynamics (5), where each stimulus was implemented as an extra current for a desired period of time.

## 3    Sequence with intervening stimuli

In order to study delay match-to-sample tasks with intervening stimuli [5, 6], the module to be identified with IT was sequentially stimulated with external currents proportional to some of the stored patterns with a delay between them. To take into account the large fraction of PF neurons with non-selective responses to the visual stimuli (which may be involved in other aspects of the task different from the identification of the stimuli), and since the neurons in our modules are, by definition, stimulus selective (although they are probably connected to the non-selective neurons) a constant, non-selective current of the same intensity as the selective input to the IT module was applied (during the same time) equally to all sub-populations of the PF module. The external current to the IT module was stimulus selective because the fraction of IT neurons with non-selective responses to the visual stimuli is very small [6]. The results can be seen in Figure 1 where the sequence **ABA** with **A** as the sample stimulus and **B** as a non-matching stimulus has been studied. The values of the model parameters are listed in the caption. In Figure 1a, the mean firing rates of the foreground populations of patterns $A_{IT}$ and $B_{IT}$ of the IT module have been plotted as a function of time. The main result is that, as observed in the experiments, the delay activity in the IT module is determined by the last stimulus presented. The delay activity provoked by a given stimulus is disrupted by the next, unless it corresponds to the same stimulus, in which case the effect of the stimulus is to increase the firing rate of the neurons in its foreground. We have checked that no noticeable effects occur if more non-matching stimuli are presented (they are all equivalent with respect to the sample) or if a non-match stimulus is repeated.

If the coupling $g$ between the modules is weak enough [12] the behavior in the PF module is different. This can be seen in Figure 1b, where the time evolution of the mean firing rates of the foreground of the two associated patterns $A_{PF}$ and $B_{PF}$ stored in the PF module are shown. In agreement with the findings of Desimone and colleagues, the neurons in the PF module remain correlated with the sample for the whole trial, despite the non-selective signal received by *all* PF neurons (not only those in the foreground of the sample) and the fact that the selective current from the IT module tends to activate the pattern associated with the *current* stimulus.

Desimone and colleagues [5, 6] report that the response of some neurons (not necessarily those with sample selective delay activity or with stimulus selective responses) in both IT and PF cortex to some stimuli, is larger if those stimuli are matches in their trials than if the same stimuli are non-matches. This has been denoted as *match enhancement*. In the present scenario the explanation is straightforward: when a stimulus is a non-match, IT and PF are in different states and therefore send inconsistent signals to each other. The firing rate of the neurons of each module is maintained in that case solely by the contribution to the total current coming from the recurrent collaterals. On the other hand, when the stimulus is the match, both modules find themselves in states associated in the synapses

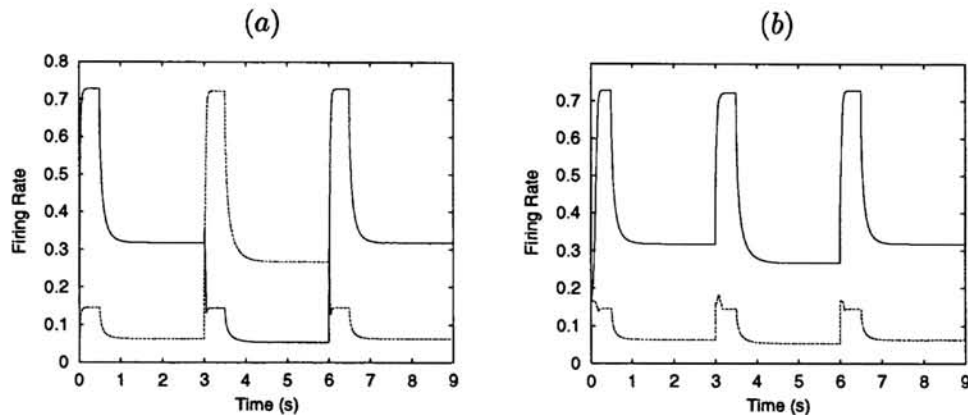

Figure 1: (*a*) Mean rates in the foreground of patterns $A_{IT}$ (solid line) and $B_{IT}$ (dashed line) in the IT module as a function of time. (*b*) Same but for patterns $A_{PF}$ and $B_{PF}$ of the PF module. Model parameters are $G = 1.3$, $\theta = 10^{-3}$, $f = 0.2$, $g = 10^{-2}$, $h = 0.13$. Stimuli are presented during $500\ ms$ at seconds 0, 3, and 6 following the sequence **ABA**.

between the neurons connecting them, PF because it has remained that way the whole trial, and IT because it is driven by the current stimulus. When this happens, the contribution to the total current from the recurrent collaterals and from the long range afferents add up consistently, and the firing rate increases. In order for this explanation to hold there should be a correlation between the top-down input from PF and the sensory bottom-up signal to IT. Indeed, experimental evidence for such a correlation has very recently been found [14]. This is an important experimental finding which supports our theory.

Looking at Figure 1, one sees that the effect is not evident in the model during the time of stimulus presentation, which is the period where it has been reported. The effect is, in fact, present, although its magnitude is too small to be noticeable in the figure. We would argue, however, that this quantitative difference is an artifact of the model. This is because the enhancement effect is very noticeable on the *delay* periods, where essentially the same neurons are active as during the stimulus presentations (i.e., where the same correlations between the top-down and bottom-up signals exist) but with lower firing rates. During stimulus presentations the firing rates are closer to the saturation regime, and therefore the dynamical response range of the neurons is largely reduced.

## 4 Memory guided attention

To test the differential response of cells as a function of the contents of memory, we have followed [7, 8] and studied a sub-population of IT cells which are simultaneously in the foreground of one of the patterns ($A_{IT}$) and in the background of another ($B_{IT}$) in the *same* conditions as the previous section (*same* model parameters). In Figure 2a the response of this sub-population as a function of time has been plotted in two different situations. In the first one, the effective stimulus $A_{IT}$ was shown first (throughout this section non selective stimulation of PF proceeded as in the last section) and after a delay, a stimulus array equal to the sum of $A_{IT}$ and $B_{IT}$ was presented. The second situation is exactly equal, except for the fact that the cue stimulus shown first was the ineffective stimulus $B_{IT}$. The response of the *same* sub-population to the *same* stimulus array is totally different and determined by the cue stimulus: If the sub-population is in the background of the cue, its response is null during the trial except for the initial period of the presentation of the array. In accordance with the experimental observations [7, 8], its response grows initially (as one would expect, since during the array presentation time, stimulation is symmetric with

respect of **A** and **B**) but is later suppressed by the top-down signal being sent by the PF module. This suppression provides a clear example of a situation in which the contents of memory (in the form of an active PF activity state) are explicitly gating the access of sensory information to IT, implementing a non-spatial attentional mechanism.

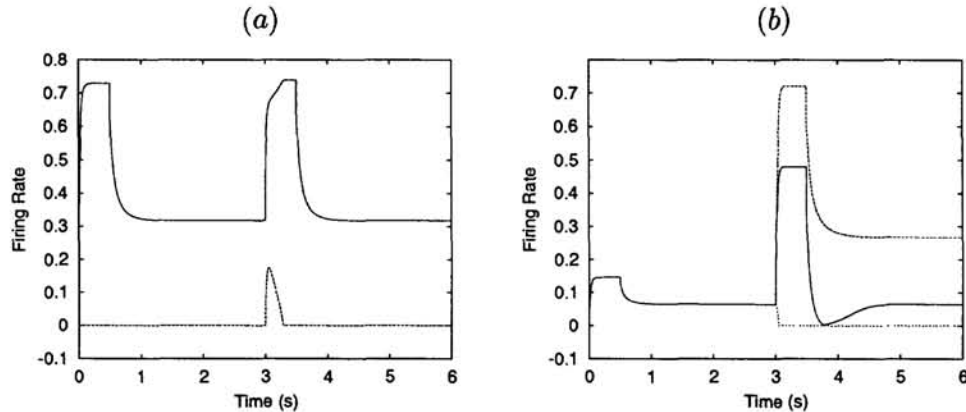

Figure 2: (*a*) Mean rates as a function of time in IT neurons which are both in the foreground $A_{IT}$ and in the background of $B_{IT}$ when the cue stimulus is $A_{IT}$ (solid line) or $B_{IT}$ (dashed line). (*b*) Mean rates of the same neurons when $C_{IT}$ is the cue stimulus and the array is $A_{IT}$ alone (long dashed line), $B_{IT}$ alone (short dashed line) or the sum of $A_{IT}$ and $B_{IT}$ (solid line). Cue present until 500 *ms*. Array present from 3000 *ms* to 3500 *ms*. Model parameters as in Figure 1

In the model, the PF module remains in a state correlated with the cue during the whole trial (to our knowledge there are no measurements of PF activity during memory guided attention tasks) and therefore provides a persistent signal 'in the direction' of the cue which *biases* the competition between $A_{IT}$ and $B_{IT}$ established at the onset of the array. This is how the gating mechanism is implemented. The competitive interactions between the stimuli in the array are studied in Figure 2b, which is an emulation of the *target-absent* trials of [8]. In this figure, the same sub-population is studied under situations in which the cue stimulus is not present in the array (another one of the stored patterns, i.e. $C_{IT}$) The three curves correspond to different arrays: The effective stimulus alone, the ineffective stimulus alone, and a sum of the two as in the previous experiment. In all three, the PF module remains in a sustained activity state correlated with $C_{IT}$ the whole trial and therefore, since the patterns are independent, the signal it sends to IT is symmetric with respect of **A** and **B**. Thus, the response of the sub-population during the array is in this case unbiased, and the effect of the competitive interactions can be isolated. The result is that, as observed experimentally, the response to the complex array is intermediate between the one to the effective stimulus alone and the one to the ineffective stimulus alone. The nature of the competition in an attractor network like the one under study here is based on the fact that complex stimulus combinations are not stored in the recurrent collaterals of each module. These connections tend to stabilize the individual patterns which, being independent, tend to cancel each other when presented together. After the array is presented, the state of the IT module, which is correlated with $C_{IT}$ in the initial delay, becomes correlated with $A_{IT}$ or $B_{IT}$ if they are presented alone. When the array contains both of them in a symmetric fashion, since the sum of the patterns is not a stored pattern itself, the IT module remains correlated with pattern $C_{IT}$ due to the signal from the PF module.

## 5  Discussion

We have proposed a toy model consisting of two reciprocally connected attractor modules which reproduces nicely experimental observations regarding intra-trial data in delay match-to-sample and memory guided attention experiments in which the interaction between IT and PF cortex is relevant. Several important issues are taken into account in the model: a complex interaction between the PF and IT modules resultant from the association of frequent patterns of activity in both modules, delay activity states in each module which exert mutually modulatory influences on each other, and a common substrate (we emphasize that the results on Sections 3 and 4 where obtained with exactly the same model parameters, just by changing the type of task) for the explanation of apparently diverse phenomena. Perception is clearly an active process which results from the complex interactions between past experience and incoming sensory information. The main goal of this model was to show that a very simple associational (Hebbian) pattern of connectivity between a perceptual module and a 'working memory' module can provide the basic ingredients needed to explain coherently different experimentally found neural mechanisms related to this process. The model has clear limitations in terms of 'biological realism' which will have to be improved in order to use it to make quantitative predictions and comparisons, and does not provide a complete an exhaustive account of the very complex and diverse phenomena in which temporo-frontal interactions are relevant (there is, for example, the issue of how to reset PF activity in between trials [15]). However, it is precisely the simplicity of the mechanism it provides and the fact that it captures the essential features of the experiments, *despite* being so simple, what makes it likely that it will remain relevant after being refined.

### Acknowledgements

This work was funded by a Spanish grant PB96-0047. We acknowledge the Max Planck Institute for Physics of Complex Systems in Dresden, Germany, for the hospitality received by A.R. and N.P. during the meeting held there from March 1 to 26, 1999.

## References

[1]  J. M. Fuster. *Memory in the cerebral cortex.* Cambridge, MA: MIT Press (1995)

[2]  D. J. Amit. *Behavioral and Brain Sciences* **18**, 617-657 (1995)

[3]  G. C. Baylis & E. T. Rolls. *Exp. Brain Res.* **65**, 614-622 (1987)

[4]  E. K. Miller, L. Li & R. Desimone. *J. Neurosci.* **13**, 1460-1478 (1993)

[5]  E. K. Miller & R. Desimone. *Science* **263**, 520-522 (1994)

[6]  E. K. Miller, C. A. Erickson & R. Desimone. *J. Neurosci.* **16**, 5154–5167 (1996)

[7]  L. Chelazzi, E. K. Miller, J. Duncan & R. Desimone. *Nature* **363**, 345-347 (1993)

[8]  L. Chelazzi, J. Duncan, E. K. Miller & R. Desimone. *J. Neurophysiol.* **80**, 2918-2940 (1998)

[9]  R. Kuhn. In *Statistical Mechanics of Neural Networks.* (ed. L. Garrido), 19-32. Berlin: Springer-Verlag (1990)

[10]  D. J. Amit & M. V. Tsodyks. *Network* **2**, 259-273 (1991)

[11]  A. Renart, N. Parga & E. T. Rolls. *Neural Computation* **11**, 1349-1388 (1999).

[12]  A. Renart, N. Parga & E. T. Rolls. *Network* **10**, 237-255 (1999).

[13]  M. Mezard, G. Parisi & M. Virasoro. *Spin glass theory and beyond.* Singapore: World Scientific (1987)

[14]  H. Tomita, M Ohbayashi, K. Nakahara, I. Hasegawa & Y. Miyashita. *Nature* **401**, 699-703 (1999)

[15]  D. Durstewitz, M. Kelc & O. Güntürkün. *J. Neurosci.* **19**, 2807-2822 (1999)